# Improved Silicon Cochlea
# using
# Compatible Lateral Bipolar Transistors

**André van Schaik, Eric Fragnière, Eric Vittoz**
MANTRA Center for Neuromimetic Systems
Swiss Federal Institute of Technology
CH-1015 Lausanne
email: vschaik@di.epfl.ch

## Abstract

Analog electronic cochlear models need exponentially scaled filters. CMOS Compatible Lateral Bipolar Transistors (CLBTs) can create exponentially scaled currents when biased using a resistive line with a voltage difference between both ends of the line. Since these CLBTs are independent of the CMOS threshold voltage, current sources implemented with CLBTs are much better matched than current sources created with MOS transistors operated in weak inversion. Measurements from integrated test chips are shown to verify the improved matching.

## 1. INTRODUCTION

Since the original publication of the "analog electronic cochlea" by Lyon and Mead in 1988 [1], several other analog VLSI models have been proposed which try to capture more of the details of the biological cochlear function [2],[3],[4]. In spite of the differences in their design, all these models use filters with exponentially decreasing cut-off frequencies. This exponential dependency is generally obtained using a linear decreasing voltage on the gates of MOS transistors operating in weak-inversion. In weak-inversion, the drain current of a saturated MOS transistor depends exponentially on its gate voltage. The linear decreasing voltage is easily created using a resistive polysilicon line; if there is a voltage difference between the two ends of the line, the voltage on the line will decrease linearly all along its length.

The problem of using MOS transistors in weak-inversion as current sources is that their drain currents are badly matched. An RMS mismatch of 12% in the drain current of two identical transistors with equal gate and source voltages is not exceptional [5], even when sufficient precautions, such as a good layout, are taken. The main cause of this mismatch is a variation of the threshold voltage between the two transistors. Since the threshold voltage and its variance are technology parameters, there is no good way to reduce the mismatch once the chip has been fabricated.

One can avoid this problem using Compatible Lateral Bipolar Transistors (CLBTs) [6] for the current sources. They can be readily made in a CMOS substrate, and their collector current also depends exponentially on their base voltage, while this current is completely independent of the CMOS technology's threshold voltage. The remaining mismatch is due to geometry mismatch of the devices, a parameter which is much better controlled than the variance of the threshold voltage. Therefore, the use of CLBTs can yield a large improvement in the regularity of the spacing of the cochlear filters. This regularity is especially important in a cascade of filters like the cochlea, since one filter can distort the input signal of all the following filters.

We have integrated an analog electronic cochlea as a cascade of second-order low-pass filters, using CLBTs as exponentially scaled current sources. The design of this cochlea is based on the silicon cochlea described in [7], since a number of important design issues, such as stability, dynamic range, device mismatch and compactness, have already been addressed in this design. In this paper, the design of [7] is briefly presented and some remaining possible improvements are identified. These improvements, notably the use of Compatible Lateral Bipolar Transistors as current sources, a differentiation that does not need gain correction and temperature independent biasing of the cut-off frequency, are then discussed in more detail. Finally, measurement results of a test chip will be presented and compared to the design without CLBTs.

## 2. THE ANALOG ELECTRONIC COCHLEA

The basic building block for the filters in all analog electronic cochlear models is the transconductance amplifier, operated in weak inversion. For input voltages smaller than about 60 mV$_{pp}$, the amplifier can be approximated as a linear transconductance:

$$I_{Out} = g_m(V_{In+} - V_{In-})\tag{1}$$

with transconductance $g_m$ given by:

$$g_m = \frac{I_0}{2nU_T}\tag{2}$$

where $I_0$ is the bias current, n is the slope factor, and the thermal voltage $U_T = kT/q = 25.6$ mV at room temperature.

This linear range is usually the input range used in the cochlear filters, yielding linear filters. In [7], a transconductance amplifier having a wider linear input range is proposed. This allows larger input signals to be used, up to about 140 mVpp. Furthermore, the wide range transconductance amplifier can be used to eliminate the large-signal instability shown to be present in the original second-order section [7]. This second-order section will be discussed in more detail in section 3.2.

The traditional techniques to improve matching [5], as for instance larger device sizes for critical devices and placing identical devices close together with identical orientation, are also discussed in [7] with respect to the implementation of the cochlear filter cascade. The transistors generating the bias current $I_0$ of the transconductance amplifiers in the second-order sections were identified as the most critical devices, since they have the largest effect on the cut-off frequency and the quality factor of each section. Therefore, extra area had to be devoted to these bias transistors. A further improvement is obtained in [7] by using a single resistive line to bias both the transconductance amplifiers controlling the cut-off frequency and the transconductance amplifier controlling the quality factor. The quality factor Q is then changed by varying the source of the transistor which biases the Q control amplifier. Instead of using two tilted resistive lines, this scheme uses only one tilted resistive line and a non-tilted Q control line, and therefore doesn't need to rely on an identical tilt on both resistive lines.

## 3. IMPROVED ANALOG ELECTRONIC COCHLEA

The design discussed in the previous section already showed a substantial improvement over the first analog electronic cochlea by Lyon and Mead. However, several improvements remain possible.

### 3.1 $V_T$ VARIATION

The bias transistors have been identified as the major source of mismatch of the cochlea's parameters. This mismatch is mainly due to variation of the threshold voltage $V_T$ of the MOS transistors. Since the drain current of a saturated MOS transistor in weak-inversion depends exponentially on the difference between its gate-source voltage and its threshold voltage, small variations in $V_T$ introduce large variations in the drain current of these transistors, and since both the cut-off frequency and the quality factor of the filters are proportional to these drain currents, large parameter variations are generated by small $V_T$ variations. This problem can be circumvented by the use of CMOS Compatible Lateral Bipolar transistors as bias transistors.

A CMOS Compatible Lateral Bipolar Transistor is obtained if the drain or source junction of a MOS transistor is forward-biased in order to inject minority carriers into the local substrate. If the gate voltage is negative enough (for an n-channel device), then no current can flow at the surface and the operation is purely bipolar [6]. Fig. 1 shows the major flows of current carriers in this mode of operation, with the source, drain and well terminals renamed emitter E, collector C and base B.

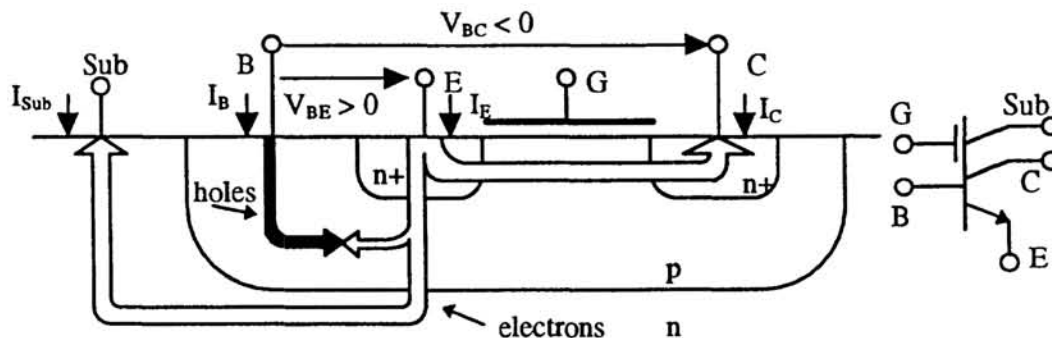

Fig. 1. : Bipolar operation of the MOS transistor : carrier flows and symbol.

Since there is no p+ buried layer to prevent injection to the substrate, this lateral npn bipolar transistor is combined with a vertical npn. The emitter current $I_E$ is thus split into a base current $I_B$, a lateral collector current $I_C$ and a substrate collector current $I_{Sub}$. Therefore, the common-base current gain $\alpha = -I_C/I_E$ cannot be close to 1. However, due to the very small rate of recombination inside the well and to the high emitter efficiency, the common-emitter current gain $\beta = I_C/I_B$ can be large. Maximum values of $\alpha$ and $\beta$ are obtained in concentric structures using a minimum size emitter surrounded by the collector and a minimum lateral base width.

For $V_{CE} = V_{BE}-V_{BC}$ larger than a few hundred millivolts, this transistor is in active mode and the collector current is given, as for a normal bipolar transistor, by

$$I_C = I_{Sb}\, e^{\dfrac{V_{BE}}{U_T}} \qquad\qquad (4)$$

where $I_{Sb}$ is the specific current in bipolar mode, proportional to the cross-section of the emitter to collector flow of carriers. Since $I_C$ is independent of the MOS transistor threshold voltage $V_T$, the main source of mismatch of distributed MOS current sources is suppressed, when CLBTs are used to create the current sources.

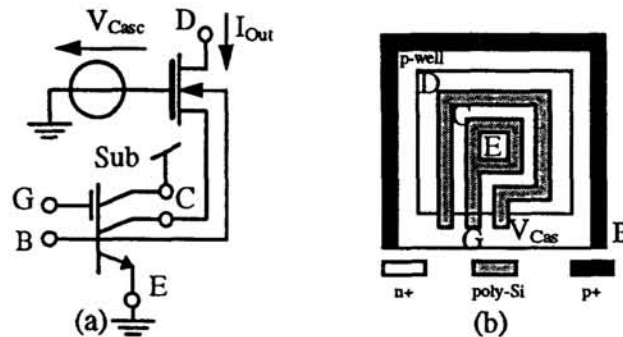

Fig. 2. CLBT cascode circuit (a) and its layout (b).

A disadvantage of the CLBT is its low Early voltage, i.e., the device has a low output resistance. Therefore, it is preferable to use a cascode circuit as shown in fig. 2. This yields an output resistance several hundred times larger than that of the single CLBT, whereas the area penalty, in a layout as shown in fig 2b, is acceptable.

Another disadvantage of the CLBTs, when biased using a resistive line, is their base current, which introduces an additional voltage drop on the resistive line. However, since the cut-off frequencies in the cochlea are controlled by the output current of the CLBTs and since these cut-off frequencies are relatively small, typically 20 kHz, the output current of the CLBTs will be small. If the common-emitter current gain $\beta$ is much larger than 1, the base current of these CLBTs will be very small, and the voltage error introduced by the small base currents will be negligible. Furthermore, since the cut-off frequencies of the cochlea will typically span 2 decades with an exponentially decreasing cut-off frequency from the beginning to the end, only the first few filters will have any noticeable influence on the current drawn from the resistive line.

### 3.2 DIFFERENTIATION

The stabilized second-order section of [7] uses two wide range transconductance amplifiers (A1 and A2 in fig. 3) with equal bias current and equal capacitive load, to control the cut-off frequency. A basic transconductance amplifier (A3) is used in a

feedback path to control the quality factor of the filter. The voltage $V_{out}$ at the output of each second-order stage represents the basilar membrane displacement. Since the output of the biological cochlea is proportional to the velocity of the basilar membrane, the output of each second-order stage has to be differentiated. In [7] this is done by creating a copy of the output current $I_{dif}$ of amplifier A2 at every stage. Since the voltage on a capacitor is proportional to the integral of the current onto the capacitor, $I_{dif}$ is effectively proportional to the basilar membrane velocity. Yet, with equal displacement amplitudes, velocity will be much larger for high frequencies than for low frequencies, yielding output signals with an amplitude that decreases from the beginning of the cochlea to the end. This can be corrected by normalizing $I_{dif}$ to give equal amplitude at every output. A second resistive line with identical tilt controlling the gain of the current mirrors that create the copies of $I_{dif}$ at each stage is used for this purpose in [7]. However, if using a single resistive line for the control of the cut-off frequencies and the quality factor improves the performance of the chip, the same is true for the control of the current mirror gain.

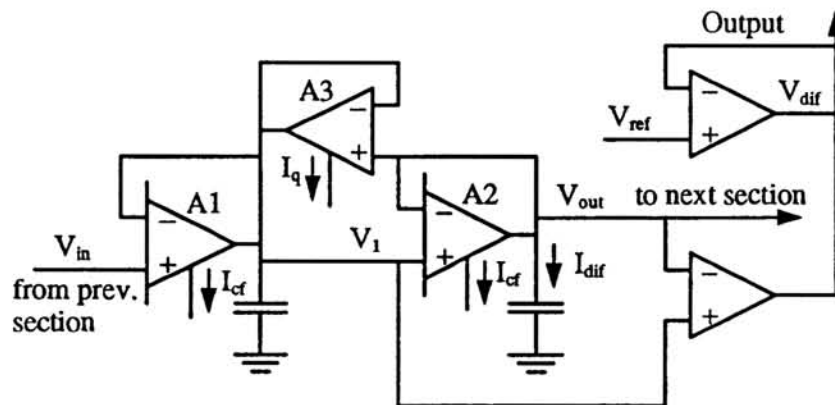

Fig. 3. One section of the cochlear cascade, with differentiator.

An alternative solution, which does not need normalization, is to take the difference between $V_{Out}$ and $V_1$ (see fig. 3). This can be shown to be equivalent to differentiating $V_{Out}$, with 0dB gain at the cut-off frequency for all stages. This can be easily done with a combination of 2 transconductance amplifiers. These amplifiers can have a large bias current, so they can also be used to buffer the cascade voltages before connecting them to the output pins of the chip, to avoid charging the cochlear cascade with the extra capacitance introduced by the output pins.

### 3.3 TEMPERATURE SENSITIVITY

The cut-off frequency of the first and the last low-pass filter in the cascade can be set by applying voltages to both ends of the resistive line, and the intermediate filters will have a cut-off frequency decreasing exponentially from the beginning to the end. Yet, if we apply directly a voltage to the ends of the resistive line, the actual cut-off frequency obtained will depend on the temperature, since the current depends exponentially on the applied voltage *normalized* to the thermal voltage $U_T$ (see(3). It is therefore better to create the voltages at both ends of the resistive line on-chip using a current biasing a CLBT with its base connected to its collector (or its drain connected to its gate if a MOS transistor is used). If this gate voltage is buffered, so that the current through the resistive line is not drawn from the input current, the bias currents of the first and last filter, and thus the cut-off frequency of all filters can be set, independent of temperature.

## 3.4 THE IMPROVED SILICON COCHLEA

The improved silicon cochlea is shown in figure 4. It uses the cochlear sections shown in figure 3, CLBTs as the bias transistors of each filter, and one resistive line to bias all CLBTs. The resistive line is biased using two bipolar current mirror structures and two voltage buffers, which allow temperature independent biasing of the cut-off frequencies of the cochlea. A similar structure is used to create the voltage source Vq to control, independent of temperature, the actual quality factor of each section. The actual bipolar current mirror implemented uses the cascode structure shown in figure 2a, however this is not shown in figure 4 for clarity.

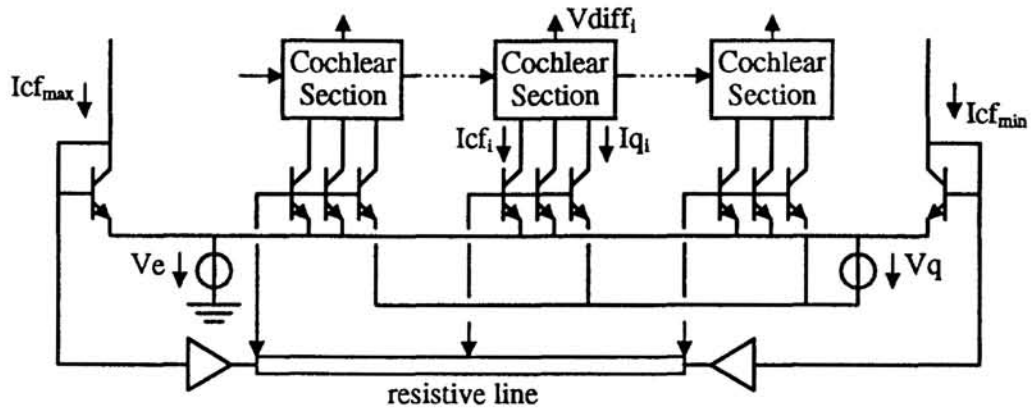

Fig 4. The improved silicon cochlea.

## 4. TEST RESULTS

The proposed silicon cochlea has been integrated using the ECPD15 technology at ES2 (Grenoble, France), containing 104 second-order stages, on a 4.77mm × 3.21mm die. Every second stage is connected to a pin, so its output voltage can be measured. In fig. 5, the frequency response curves after on-chip derivation are shown for the output taps of both the cochlea described in [7] (left), and our version (right). This clearly shows the improved regularity of the cut-off frequencies and the gain obtained using CLBTs. The drop-off in gain for the higher frequency stages (right) is a border effect, since at the beginning of the cochlea no accumulation of gain has yet taken place. In the figure on the left this is not visible, since the first nine outputs are not presented.

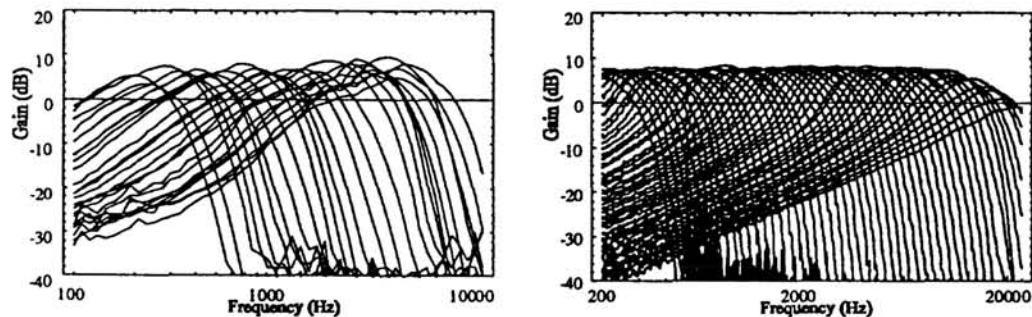

Fig.5. Measured frequency responses at the different taps.

In fig. 6 we show the cut-off frequency versus tap number of both chips. Ideally, this should be a straight line on a log-linear scale, since the cut-off frequency decreases

exponentially with tap number. This also clearly shows the improved regularity using CLBTs as current sources.

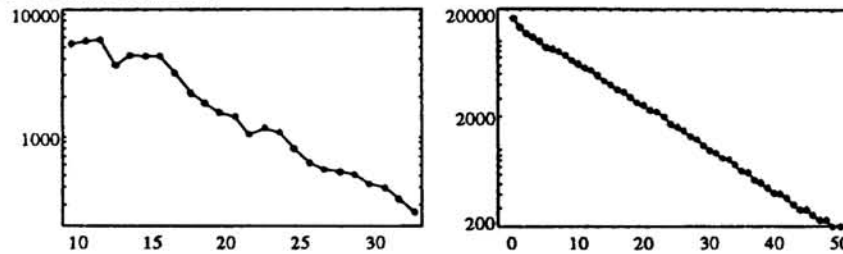

Fig.6. Cut-off frequency (Hz) versus tap number for both silicon cochleae.

## 5. CONCLUSIONS

Since the biological cochlea functions as a distributed filter, where the natural frequency decreases exponentially with the position along the basilar membrane, analog electronic cochlear models need exponentially scaled filters. The output current of a Compatible Lateral Bipolar Transistor depends exponentially on the base-emitter voltage. It is therefore easy to create exponentially scaled current sources using CLBTs biased with a resistive polysilicon line. Because the CLBTs are insensitive to variations of the CMOS threshold voltage $V_T$, current sources implemented with CLBTs are much better matched than current sources using MOS transistors in weak inversion.

Regularity is further improved using an on-chip differentiation that does not need a second resistive line to correct its gain, and therefore doesn't depend on identical tilt on both resistive lines. Better independence of temperature can be obtained by fixing the frequency domain of the cochlea using bias currents instead of voltages.

### Acknowledgments

The authors would like to thank Felix Lustenberger for simulation and layout of the chip. We are also indebted to Lloyd Watts for allowing us to use his measurement data.

### References

[1] R.F. Lyon and C.A. Mead, "An analog electronic cochlea," *IEEE Trans. Acoust., Speech, Signal Processing*, vol. 36, pp. 1119-1134, July 1988.

[2] R.F. Lyon, "Analog implementations of auditory models," *Proc. DARPA Workshop Speech and Natural Language*. San Mateo, CA:Morgan Kaufmann, 1991.

[3] W. Liu, et. al., "Analog VLSI implementation of an auditory periphery model," *Advances Res. VLSI, Proc. 1991 Santa Cruz Conf.*, MIT Press, 1991, pp. 153-163.

[4] L. Watts, "Cochlear Mechanics: Analysis and Analog VLSI," Ph.D. thesis, California Institute of Technology, Pasadena, 1992.

[5] E. Vittoz, "The design of high-performance analog circuits on digital CMOS chips," *IEEE J. Solid-State Circuits*, vol. SC-20, pp. 657-665, June 1985.

[6] E. Vittoz, "MOS transistors operated in the lateral bipolar mode and their application in CMOS technology," *IEEE J. Solid-State Circuits*, vol. SC-24, pp. 273-279, June 1983.

[7] L. Watts, et. al., "Improved implementation of the silicon cochlea," *IEEE J. Solid-State Circuits*, vol. SC-27, pp. 692-700, May 1992.